# Conditional mean field

**Peter Carbonetto**
Department of Computer Science
University of British Columbia
Vancouver, BC, Canada V6T 1Z4
pcarbo@cs.ubc.ca

**Nando de Freitas**
Department of Computer Science
University of British Columbia
Vancouver, BC, Canada V6T 1Z4
nando@cs.ubc.ca

## Abstract

Despite all the attention paid to variational methods based on sum-product message passing (loopy belief propagation, tree-reweighted sum-product), these methods are still bound to inference on a small set of probabilistic models. Mean field approximations have been applied to a broader set of problems, but the solutions are often poor. We propose a new class of *conditionally-specified* variational approximations based on mean field theory. While not usable on their own, combined with sequential Monte Carlo they produce guaranteed improvements over conventional mean field. Moreover, experiments on a well-studied problem—inferring the stable configurations of the Ising spin glass—show that the solutions can be significantly better than those obtained using sum-product-based methods.

## 1 Introduction

Behind all variational methods for inference in probabilistic models lies a basic principle: treat the quantities of interest, which amount to moments of the random variables, as the solution to an optimization problem obtained via convex duality. Since optimizing the dual is rarely an amelioration over the original inference problem, various strategies have arisen out of statistical physics and machine learning for making principled (and unprincipled) approximations to the objective.

One such class of techniques, *mean field theory*, requires that the solution define a distribution that factorizes in such a way that the statistics of interest are easily derived. Mean field remains a popular tool for statistical inference, mainly because it applies to a wide range of problems. As remarked by Yedidia in [17], however, mean field theory often imposes unrealistic or questionable factorizations, leading to poor solutions. Advances have been made in improving the quality of mean field approximations [17, 22, 26], but their applicability remains limited to specific models. Bethe-Kikuchi approximations overcome some of the severe restrictions on factorizability by decomposing the entropy according to a junction graph [1], for which it is well established that generalized belief propagation updates converge to the stationary points of the resulting optimization problem (provided they converge at all). Related variational approximations based on convex combinations of tree-structured distributions [24] have the added advantage that they possess a unique global optimum (by contrast, we can only hope to discover a local minimum of the Bethe-Kikuchi and mean field objectives). However, both these methods rely on tractable sum-product messages, hence are limited to Gaussian Markov random fields or discrete random variables. Expectation propagation projections and Monte Carlo approximations to the sum-product messages get around these limitations, but can be unsuitable for dense graphs or can introduce extraordinary computational costs [5, 23]. Thus, there still exist factorized probabilistic models, such as sigmoid belief networks [21] and latent Dirichlet allocation [5], whereby mean field remains to date the tractable approximation of choice.

Several Monte Carlo methods have been proposed to correct for the discrepancy between the factorized variational approximations and the target distribution. These methods include importance sampling [8, 14] and adaptive Markov Chain Monte Carlo (MCMC) [6]. However, none of these techniques scale well to general, high-dimensional state spaces because the variational approxi-

mations tend to be too restrictive when used as a proposal distribution. This is corroborated by experimental results in those papers as well as theoretical results [20]. We propose an entirely new approach that overcomes the problems of the aforementioned methods by constructing a sequence of variational approximations that converges to the target distribution. To accomplish this, we derive a new class of *conditionally-specified* mean field approximations, and use sequential Monte Carlo (SMC) [7] to obtain samples from them. SMC acts as a mechanism to migrate particles from an easy-to-sample distribution (naive mean field) to a difficult-to-sample one (the distribution of interest), through a sequence of artificial distributions. Each artificial distribution is a *conditional mean field* approximation, designed in such a way that it is at least as sensible as its predecessor because it recovers dependencies left out by mean field. Sec. 4 explains these ideas thoroughly.

The idea of constructing a sequence of distributions has a strong tradition in the literature, dating back to work on simulating the behaviour of polymer chains [19] and counting and integration problems [12]. Recent advances in stochastic simulation have allowed practitioners to extend these ideas to general probabilistic inference [7, 11, 15]. However, very little is known as to how to come up with a good sequence of distributions. Tempering is perhaps the most widely used strategy, due to its ease of implementation and intuitive appeal. At early stages, high global temperatures smooth the modes and allow easy exploration of the state space. Afterward, the temperature is progressively cooled until the original distribution is recovered. The problem is that the variance of the importance weights tends to degenerate around a system's critical range of temperatures, as observed in [9]. An entirely different approach is to remove constraints (or factors) from the original model, then incrementally reintroduce them. This has been a fruitful approach for approximate counting [12], simulation of protein folding, and inference in the Ising model [9]. If, however, a reintroduced constraint has a large effect on the distribution, the particles may again rapidly deterioriate.

We limit our study to the Ising spin glass model [16]. Ernst Ising developed his model in order to explain the phenomenon of "spontaneous magnetization" in magnets. Here, we use it as a test bed to investigate the viability or our proposed algorithm. Our intent is *not* to design an algorithm tuned to sampling the states of the Ising model, but rather to tackle factorized graphical models with arbitrary potentials. Conditional mean field raises many questions, and since we can only hope to answer some in this study, the Ising model represents a respectable first step. We hint at how our ideas might generalize in Sec. 6.

The next two sections serve as background for the presentation of our main contribution in Sec. 4.

## 2  Mean field theory

In this study, we restrict our attention to random vectors $X = (X_1, \ldots, X_n)^T$, with possible configurations $x = (x_1, \ldots, x_n)^T \in \Omega$, that admit a distribution belonging to the standard exponential family [25]. A member of this family has a probability density of the form

$$p(x; \theta) = \exp\left\{\theta^T \phi(x) - \Psi(\theta)\right\}, \tag{1}$$

where $\theta$ is the canonical vector of parameters, and $\phi(x)$ is the vector of *sufficient statistics* [25]. The log-partition function $\Psi(\theta)$ ensures that $p(x; \theta)$ defines a valid probability density, and is given by

$$\Psi(\theta) = \log \int \exp\left\{\theta^T \phi(x)\right\} dx.$$

Denoting $\mathbb{E}_\pi\{f(X)\}$ to be the expected value of a function $f(x)$ with respect to distribution $\pi$, Jensen's inequality states that $f(\mathbb{E}_\pi\{X\}) \leq \mathbb{E}_\pi\{f(X)\}$ for any convex function $f(x)$ and distribution $\pi$ on $X$. Using the fact that $-\log(x)$ is convex, we obtain the variational lower bound

$$\Psi(\theta) = \log \mathbb{E}_{p(\cdot;\alpha)}\left\{\frac{\exp(\theta^T \phi(X))}{p(X;\alpha)}\right\} \geq \theta^T \mu(\alpha) - \int p(x; \alpha) \log p(x; \alpha)\, dx, \tag{2}$$

where the *mean statistics* are defined by $\mu(\alpha) \equiv \mathbb{E}_{p(\cdot;\alpha)}\{\phi(X)\}$. The second term on the right-hand side of (2) is the Boltzmann-Shannon entropy of $p(x; \alpha)$, which we denote by $H(\alpha)$. Clearly, some lower bounds of the form (2) are better than others, so the optimization problem is to find a set of parameters $\alpha$ that leads to the tightest bound on the log-partition function. This defines the *variational principle*. We emphasize that this lower bound holds for *any* choice of $\alpha$. A more rigorous treatment follows from analyzing the conjugate of the convex, differentiable function $\Psi(\theta)$ [25].

As it is presented here, the variational principle is of little practical use because no tractable expressions exist for the entropy and mean statistics. There do, however, exist particular choices of the

variational parameters $\alpha$ where it is possible to compute them both. We shall examine one particular set of choices, *naive mean field*, in the context of the Ising spin glass model.

At each site $i \in \{1, \ldots, n\}$, the random variable $X_i$ is defined to be $x_i = +1$ if the magnetic dipole in the "up" spin position, or $x_i = -1$ if it is "down". Each scalar $\theta_{ij}$ defines the interaction between sites $i$ and $j$. Setting $\theta_{ij} > 0$ causes attraction between spins, and $\theta_{ij} < 0$ induces repulsion. Scalars $\theta_i$ define the effect of the external magnetic field on the energy of the system. We use the undirected labelled graph $G = (V, E)$, where $V = \{1, \ldots, n\}$, to represent the conditional independence structure of the probability measure (there is no edge between $i$ and $j$ if and only if $X_i$ and $X_j$ are conditionally independent given values at all other points of the graph). Associating singleton factors with nodes of $G$ and pairwise factors with its edges, and setting the entries of the sufficient statistics vector to be $x_i, \forall\, i \in V$ and $x_i x_j, \forall\, (i,j) \in E$, we can write the probability density as

$$p(x; \theta) = \exp \left\{ \sum_{i \in V} \theta_i x_i + \sum_{(i,j) \in E} \theta_{ij} x_i x_j - \Psi(\theta) \right\}. \tag{3}$$

The corresponding variational lower bound on the log-partition function $\Psi(\theta)$ then decomposes as

$$F(\alpha) \equiv \sum_{i \in V} \theta_i \mu_i(\alpha) + \sum_{(i,j) \in E} \theta_{ij} \mu_{ij}(\alpha) + H(\alpha), \tag{4}$$

where $\mu_i(\alpha)$ and $\mu_{ij}(\alpha)$ are the expectations of single spins $i$ and pairs of spins $(i, j)$, respectively.

Naive mean field restricts the variational parameters $\alpha$ to belong to $\{\alpha \,|\, \forall\, (i,j) \in E, \alpha_{ij} = 0\}$. We can compute the lower bound (4) for any $\alpha$ belonging to this subset because we have tractable expressions for the mean statistics and entropy. For the Ising spin glass, the mean statistics are

$$\mu_i(\alpha) \equiv \int x_i \, p(x; \alpha)\, dx = \tanh(\alpha_i) \tag{5}$$

$$\mu_{ij}(\alpha) \equiv \int x_i \, x_j \, p(x; \alpha)\, dx = \mu_i(\alpha)\, \mu_j(\alpha), \tag{6}$$

and the entropy is derived to be

$$H(\alpha) = -\sum_{i \in V} \left( \tfrac{1 - \mu_i(\alpha)}{2} \right) \log \left( \tfrac{1 - \mu_i(\alpha)}{2} \right) - \sum_{i \in V} \left( \tfrac{1 + \mu_i(\alpha)}{2} \right) \log \left( \tfrac{1 + \mu_i(\alpha)}{2} \right). \tag{7}$$

The standard way to proceed [17, 25] is to derive coordinate ascent updates by equating the derivatives $\partial F / \partial \mu_i$ to zero and solving for $\mu_i$. Since the variables $\mu_i$ must be valid mean statistics, they are constrained to lie within an envelope known as the *marginal polytope* [25]. Alternatively, one can solve the optimization problem with respect to the unconstrained variational parameters $\alpha$. Since it is not possible to obtain the fixed-point equations by isolating each $\alpha_i$, instead one can easily derive expressions for the gradient $\nabla F(\alpha)$ and Hessian $\nabla^2 F(\alpha)$ and run a nonlinear optimization routine. This approach, as we will see, is necessary for optimizing the conditional mean field objective.

## 3 Sequential Monte Carlo

Consider a sequence of two distributions, $\pi(x)$ and $\pi^\star(x)$, where the second represents the target. Assuming familiarity with importance sampling, this will be sufficient to explain key concepts underlying SMC, and does not overwhelm the reader with subscripts. See [7] for a detailed description.

In the first step, samples $x^{(s)} \in \Omega$ are drawn from some proposal density $q(x)$ and assigned importance weights $w(x) = \pi(x)/q(x)$. In the second step, a Markov transition kernel $K^\star(x' \,|\, x)$ shifts each sample towards the target, and the importance weights $\tilde{w}(x, x')$ compensate for any failure to do so. In effect, the second step consists of extending the path of each particle onto the joint space $\Omega \times \Omega$. The unbiased importance weights on the joint space are given by

$$\tilde{w}(x, x') = \frac{\tilde{\pi}(x, x')}{\tilde{q}(x, x')} = \frac{L(x \,|\, x')\, \pi^\star(x')}{K^\star(x' \,|\, x)\, \pi(x)} \times w(x), \tag{8}$$

where $\tilde{\pi}(x, x') = L(x \,|\, x')\, \pi^\star(x')$ is the artificial distribution over the joint space, $\tilde{q}(x, x') = K^\star(x' \,|\, x)\, q(x)$ is the corresponding importance distribution, and the "backward-in-time" kernel $L(x \,|\, x')$ is designed so that it admits $\pi(x)$ as its invariant distribution. Our expectation is that $K^\star(x' \,|\, x)$ have invariant distribution $\pi^\star(x)$, though it is not required. To prevent potential particle degeneracy in the marginal space, we adopt the standard stratified resampling algorithm [13].

**Choice of backward-in-time kernel.** Mean field tends to be overconfident in its estimates (although not necessarily so). Loosely speaking, this means that if $\pi(x)$ were to be a mean field approximation,

then it would likely have lighter tails than the target distribution $\pi^\star(x)$. If we were to use a sub-optimal backward kernel [7, Sec. 3.3.2.3], the importance weights would simplify to

$$\tilde{w}(x, x') = \pi^\star(x) / \pi(x) \times w(x). \tag{9}$$

Implicitly, this is the choice of backward kernel made in earlier sequential frameworks [11, 15]. Since the mean field approximation $\pi(x)$ might very well fail to "dominate" the target $\pi^\star(x)$, the expression (9) risks having unbounded variance. This is a problem because the weights may change abruptly from one iteration to the next, or give too much importance to too few values $x$ [18]. Instead, Del Moral *et al* suggest approximating the optimal backward-in-time kernel [7, Sec. 3.3.2.1] by

$$L(x \mid x') = \frac{K^\star(x' \mid x)\,\pi(x)}{\int K^\star(x' \mid x)\,\pi(x)\,dx}. \tag{10}$$

It offers some hope because the resulting importance weights on the joint space, following (8), are

$$\tilde{w}(x, x') = \frac{\pi^\star(x')}{\int K^\star(x' \mid x)\,\pi(x)\,dx} \times w(x). \tag{11}$$

If the transition kernel increases the mass of the proposal in regions where $\pi(x)$ is weak relative to $\pi^\star(x)$, the backward kernel (10) will rectify the problems caused by an overconfident proposal.

**Choice of Markov transition kernel.** The drawback of the backward kernel (10) is that it limits the choice of transition kernel $K^\star(x' \mid x)$, a crucial ingredient to a successful SMC simulation. For instance, we can't use the Metropolis-Hastings algorithm because its transition kernel involves an integral that does not admit a closed form [18]. One transition kernel which fits our requirements and is widely applicable is a mixture of kernels based on the random-scan Gibbs sampler [18]. Denoting $\delta_y(x)$ to be the Dirac measure at location $y$, the transition kernel with invariant distribution $\pi^\star(x)$ is

$$K^\star(x' \mid x) = \sum_k \rho_k \pi^\star(x'_k \mid x_{-k})\,\delta_{x_{-k}}(x'_{-k}), \tag{12}$$

where $\pi(x_k \mid x_{-k})$ is the conditional density of $x_k$ given values at all other sites, and $\rho_k$ is the probability of shifting the samples according to the Gibbs kernel at site $k$. Following (11) and the identity for conditional probability, we arrive at the expression for the importance weights,

$$\tilde{w}(x, x') = \frac{\pi^\star(x')}{\pi(x')} \left\{ \sum_k \rho_k \frac{\pi^\star(x'_k \mid x'_{-k})}{\pi(x'_k \mid x'_{-k})} \right\}^{-1} \times w(x). \tag{13}$$

**Normalized estimator.** For almost all problems in Bayesian analysis (and certainly the one considered in this paper), the densities are only known up to a normalizing constant. That is, only $f(x)$ and $f^\star(x)$ are known pointwise, where $\pi(x) = f(x)/Z$ and $\pi^\star(x) = f^\star(x)/Z^\star$. The normalized importance sampling estimator [18] yields (asymptotically unbiased) importance weights $\tilde{w}(x, x') \propto \hat{w}(x, x')$, where the *unnormalized importance weights* $\hat{w}(x, x')$ in the joint space remain the same as (13), except that we substitute $\pi(x)$ for $f(x)$, and $\pi^\star(x)$ for $f^\star(x)$. The normalized estimator can recover a Monte Carlo estimate of the normalizing constant $Z^\star$ via the recursion

$$Z^\star \approx Z \times \sum_s \hat{w}^{(s)}, \tag{14}$$

provided we already have a good estimate of $Z$ [7].

## 4 Conditional mean field

We start with a partition $R$ (equivalence relation) of the set of vertices $V$. Elements of $R$, which we denote with the capital letters $A$ and $B$, are disjoint subsets of $V$. Our strategy is to come up with a good naive mean field approximation to the conditional density $p(x_A \mid x_{-A}; \theta)$ for every equivalence class $A \in R$, and then again for every configuration $x_{-A}$. Here, we denote $x_A$ to be the configuration $x$ restricted to set $A \subseteq V$, and $x_{-A}$ to be the restriction of $x$ to $V \setminus A$. The crux of the matter is that for any point $\alpha$, the functions $p(x_A \mid x_{-A}; \alpha)$ *only* represent valid conditional densities if they correspond to some unique joint, as discussed in [2]. Fortunately, under the Ising model the terms $p(x_A \mid x_{-A}; \alpha)$ represent valid conditionals for *any* $\alpha$. What we have is a slight generalization of the auto-logistic model [3], for which the joint is always known. As noted by Besag, "although this is derived classically from thermodynamic principles, it is remarkable that the Ising model follows necessarily as the very simplest non-trivial binary Markov random field [4]."

Conditional mean field forces each conditional $p(x_A \mid x_{-A}; \alpha)$ to decompose as a product of marginals $p(x_i \mid x_{-A}; \alpha)$, for all $i \in A$. As a result, $\alpha_{ij}$ must be zero for every edge $(i, j) \in E(A)$, where we define $E(A) \equiv \{(i, j) \mid i \in A, j \in A\}$ to be the set of edges contained by the vertices in subset $A$. Notice that we have a set of free variational parameters $\alpha_{ij}$ defined on the edges $(i, j)$ that straddle subsets of the partition. Formally, these are the edges that belong to $C_R \equiv \{(i, j) \mid \forall A \in R, (i, j) \notin E(A)\}$. We call $C_R$ the set of "connecting edges".

Our variational formulation consists of competing objectives, since the conditionals $p(x_A \mid x_{-A}; \alpha)$ share a common set of parameters. We formulate the final objective function as a linear combination of *conditional objectives*. A conditional mean field optimization problem with respect to graph partition $R$ and linear weights $\lambda$ is of the form

$$\begin{aligned} \text{maximize} \quad & F_{R,\lambda}(\alpha) \equiv \sum_{A \in R} \sum_{x_{N(A)}} \lambda_A(x_{N(A)}) F_A(\alpha, x_{N(A)}) \\ \text{subject to} \quad & \alpha_{ij} = 0, \text{ for all } (i, j) \in E \setminus C_R. \end{aligned} \tag{15}$$

We extend the notion of neighbours to sets, so that $N(A)$ is the *Markov blanket* of $A$. The non-negative scalars $\lambda_A(x_{N(A)})$ are defined for every equivalence class $A \in R$ and configuration $x_{N(A)}$. Each conditional objective $F_A(\alpha, x_{N(A)})$ represents a naive mean field lower bound to the log-partition function of the conditional density $p(x_A \mid x_{-A}; \theta) = p(x_A \mid x_{N(A)}; \theta)$. For the Ising model, $F_A(\alpha, x_{N(A)})$ follows from the exact same steps used in the derivation of the naive mean field lower bound in Sec. 2, except that *we replace the joint by a conditional*. We obtain the expression

$$\begin{aligned} F_A(\alpha, x_{N(A)}) = \sum_{i \in A} \theta_i \mu_i(\alpha, x_{N(A)}) + \sum_{(i,j) \in E(A)} \theta_{ij} \mu_{ij}(\alpha, x_{N(A)}) \\ + \sum_{i \in A} \sum_{j \in (N(i) \cap N(A))} \theta_{ij} x_j \mu_i(\alpha, x_{N(A)}) + H_A(\alpha, x_{N(A)}), \end{aligned} \tag{16}$$

with the *conditional mean statistics* for $i \in A, j \in A$ given by

$$\mu_i(\alpha, x_{N(A)}) \equiv \int x_i \, p(x_A \mid x_{N(A)}; \alpha) \, dx = \tanh\left(\alpha_i + \sum_{j \in (N(i) \cap N(A))} \alpha_{ij} x_j\right) \tag{17}$$

$$\mu_{ij}(\alpha, x_{N(A)}) \equiv \int x_i \, x_j \, p(x_A \mid x_{N(A)}; \alpha) \, dx = \mu_i(\alpha, x_{N(A)}) \mu_j(\alpha, x_{N(A)}). \tag{18}$$

The entropy is identical to (7), with the mean statistics replaced with their conditional counterparts. Notice the appearance of the new terms in (16). These terms account for the interaction between the random variables on the border of the partition. We can no longer optimize $\mu$ following the standard approach; we cannot treat the $\mu_i(\alpha, x_{N(A)})$ as independent variables for all $x_{N(A)}$, as the solution would no longer define an Ising model (or even a valid probability density, as we discussed). Instead, we optimize with respect to the parameters $\alpha$, taking derivatives $\nabla F_{R,\lambda}(\alpha)$ and $\nabla^2 F_{R,\lambda}(\alpha)$.

We have yet to address the question: how to select the scalars $\lambda$? It stands to reason that we should place greater emphasis on those conditionals that are realised more often, and set $\lambda_A(x_{N(A)}) \propto p(x_{N(A)}; \theta)$. Of course, these probabilities aren't available! Equally problematic is the fact that (15) may involve nearly as many terms as there are possible worlds, hence offering little improvement over the naive solution. As it turns out, a greedy choice resolves both issues. Supposing that we are at some intermediate stage in the SMC algorithm (see Sec. 4.1), a greedy but not unreasonable choice is to set $\lambda_A(x_{N(A)})$ to be the current Monte Carlo estimate of the marginal $p(x_{N(A)}; \theta)$,

$$\lambda_A(x_{N(A)}) = \sum_s w^{(s)} \delta_{x_{N(A)}^{(s)}}(x_{N(A)}). \tag{19}$$

Happily, the number of terms in (15) is now on the order of the number of the particles.

Unlike standard naive mean field, conditional mean field optimizes over the pairwise interactions $\alpha_{ij}$ defined on the connecting edges $(i, j) \in C_R$. In our study, we fix these parameters to $\alpha_{ij} = \theta_{ij}$. This choice is convenient for two reasons. First, the objective is separable on the subsets of the partition. Second, the conditional objective of a singleton subset has a unique maximum at $\alpha_i = \theta_i$, so any solution to (15) is guaranteed to recover the original distribution when $|R| = n$.

## 4.1   The Conditional mean field algorithm

We propose an SMC algorithm that produces progressively refined particle estimates of the mean statistics, in which conditional mean field acts in a supporting role. The initial SMC distribution is obtained by solving (15) for $R = \{V\}$, which amounts to the mean field approximation derived in Sec. 2. In subsequent steps, we iteratively solve (15), update the estimates of the mean statistics by reweighting (see (20)) and occasionally resampling the particles, then we split the partition until we cannot split it anymore, at which point $|R| = n$ and we recover the target $p(x; \theta)$. It is easy to

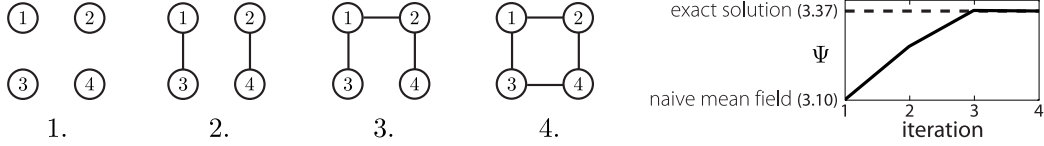

Figure 1: The graphs on the left depict the Markov properties of the conditional mean field approximations in steps 1 to 4. Graph #4 recovers the target. In the right plot, the solid line is the evolution of the estimate of the log-partition function in SMC steps 1 to 4. The dashed line is the true value.

draw samples from the initial fully-factorized distribution. It is also easy to compute its log-partition function, as $\Psi(\alpha) = \sum_{i \in V} \log(2 \cosh(\alpha_i))$. Note that this estimate is *not* a variational lower bound.

Let's now suppose we are at some intermediate step in the algorithm. We currently have a particle estimate of the $R$-partition conditional mean field approximation $p(x; \alpha)$ with samples $x^{(s)}$ and marginal importance weights $w^{(s)}$. To construct the next artificial distribution $p(x; \alpha^\star)$ in the sequence, we choose a finer partitioning of the graph, $R^\star$, set the weights $\lambda^\star$ according to (19), and use a nonlinear solver to find a local minimum $\alpha^\star$ to (15). The solver is initialized to $\alpha_i^\star = \theta_i$. We require that the new graph partition satisfy that for every $B \in R^\star$, $B \subseteq A$ for some $A \in R$. In this manner, we ensure that the sequence is progressing toward the target (provided $R \neq R^\star$), and that it is always possible to evaluate the importance weights. It is not understood how to tractably choose a good sequence of partitions, so we select them in an arbitrary manner. Next, we use the random-scan Gibbs sampler (12) to shift the particles toward the new distribution, where the Gibbs sites $k$ correspond to the subsets $B \in R^\star$. We set the mixture probabilities of the Markov transition kernel to $\rho_B = |B|/n$. Following (13), the expression for the unnormalized importance weights is

$$\hat{w}(x, x') = \frac{\exp\left(\sum_i \alpha_i^\star x_i' + \sum_{(i,j)} \alpha_{ij}^\star x_i' x_j'\right)}{\exp\left(\sum_i \alpha_i x_i' + \sum_{(i,j)} \alpha_{ij} x_i' x_j'\right)} \left\{ \sum_{B \in R^\star} \rho_B \prod_{i \in B} \frac{\pi(x_i' \mid x_{N(B)}'; \alpha^\star)}{\pi(x_i' \mid x_{N(A)}'; \alpha)} \right\}^{-1} \times w(x), \quad (20)$$

where the single-site conditionals are $\pi(x_i \mid x_{N(A)}; \alpha) = (1 + x_i \mu_i(\alpha, x_{N(A)}))/2$ and $A \in R$ is the unique subset containing $B \in R^\star$. The new SMC estimate of the log-partition function is $\Psi(\alpha^\star) \approx \Psi(\alpha) + \log \sum_s \hat{w}^{(s)}$. To obtain the particle estimate of the new distribution, we normalize the weights $\tilde{w}^{(s)} \propto \hat{w}^{(s)}$, assign the marginal importance weights $w^{(s)} \leftarrow \tilde{w}^{(s)}$, and set $x^{(s)} \leftarrow (x')^{(s)}$. We are now ready to move to the next iteration. Let's look at a small example to see how this works.

**Example.** Consider an Ising model with $n = 4$ and parameters $\theta_{1:4} = \frac{1}{10}(4, 3, -5, -2)$, $\theta_{13} = \theta_{24} = \theta_{34} = +\frac{1}{2}$ and $\theta_{12} = -\frac{1}{2}$. We assume we have enough particles to recover the distributions almost perfectly. Setting $R = \{\{1, 2, 3, 4\}\}$, the first artificial distribution is the naive mean field solution $\alpha_{1:4} = (0.09, 0.03, -0.68, -0.48)$ with $\Psi(\alpha) = 3.10$. Knowing that the true mean statistics are $\mu_{1:4} = (0.11, 0.07, -0.40, -0.27)$, and $\text{Var}(X_i) = 1 - \mu_i^2$, it is easy to see naive mean field largely underestimates the variance of the spins. In step 2, we split the partition into $R = \{\{1, 2\}, \{3, 4\}\}$, and the new conditional mean field approximation is given by $\alpha_{1:4} = (0.39, 0.27, -0.66, -0.43)$, with potentials $\alpha_{13} = \theta_{13}$, $\alpha_{24} = \theta_{24}$ on the connecting edges $C_R$. The second distribution recovers the two dependencies between the subsets, as depicted in Fig. 1. Step 3 then splits subset $\{1, 2\}$, and we get $\alpha = (0.40, 0.30, -0.64, -0.42)$ by setting $\lambda$ according to the weighted samples from step 2. Notice that $\alpha_1 = \theta_1$, $\alpha_2 = \theta_2$. Step 4 recovers the original distribution, at which point the estimate of the log-partition function comes close to the exact solution, as shown in Fig. 1. In this example, $\Psi(\alpha)$ happens to underestimate $\Psi(\theta)$, but in other examples we may get overestimates.

The random-scan Gibbs sampler can mix poorly, especially on a fine graph partition. Gradually changing the parameters with tempered artificial distributions [7, Sec. 2.3.1] $p(x; \alpha)^{1-\gamma} p(x; \alpha^\star)^\gamma$ gives the transition kernel more opportunity to correctly migrate the samples to the next distribution.

To optimize (15), we used a stable modification to Newton's method that maintains a quadratic approximation to the objective with a positive definite Hessian. In light of our experiences, a better choice might have been to sacrifice the quadratic convergence rate for a limited-memory Hessian approximation or conjugate gradient; the optimization routine was the computational bottleneck on dense graphs. Even though the solver is executed at every iteration of SMC, the separability of the objective (15) means that the computational expense decreases significantly at every iteration. To our knowledge, this is the only SMC implementation in which the next distribution in the sequence is constructed dynamically according to the particle approximation from the previous step.

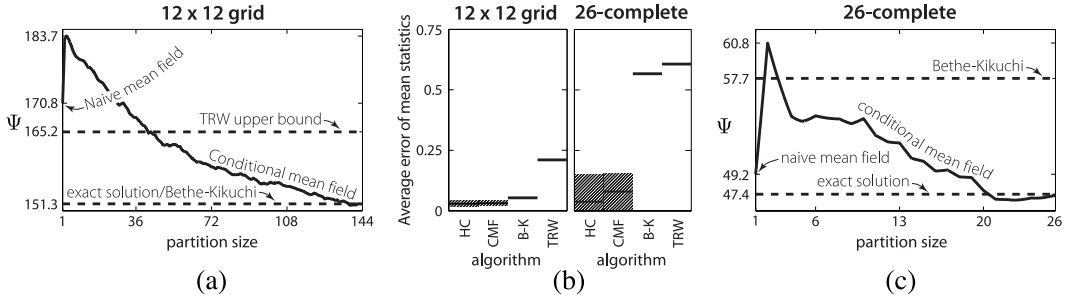

Figure 2: *(a)* Estimate of the $12 \times 12$ grid log-partition function for each iteration of SMC. *(c)* Same, for the fully-connected graph with 26 nodes. We omitted the tree-reweighted upper bound because it is way off the map. Note that these plots will vary slightly for each simulation. *(b)* Average error of the mean statistics according to the hot coupling (HC), conditional mean field algorithm (CMF), Bethe-Kikuchi variational approximation (B-K), and tree-reweighted upper bound (TRW) estimates. The maximum possible average error is 2. For the HC and CMF algorithms, 95% of the estimates fall within the shaded regions according to a sample of 10 simulations.

## 5    Experiments

We conduct experiments on two Ising models, one defined on a $12 \times 12$ grid, and the other on a fully-connected graph with 26 nodes. The model sizes approach the limit of what we can compute exactly for the purposes of evaluation. The magnetic fields are generated by drawing each $\theta_i$ uniformly from $[-1, 1]$ and drawing $\theta_{ij}$ uniformly from $\{-\frac{1}{2}, +\frac{1}{2}\}$. Both models exhibit strong and conflicting pairwise interactions, so it is expected that rudimentary MCMC methods such as Gibbs sampling will get "stuck" in local modes [9]. Our algorithm settings are as follows. We use 1000 particles (as with most particle methods, the running time is proportional to the number of particles), and we temper across successive distributions with a linear inverse temperature schedule of length 100. The particles are resampled when the effective sample size [18] drops below $\frac{1}{2}$. We compare our results with the "hot coupling" SMC algorithm described in [9] (appropriately, using the same algorithm settings), and with two sum-product methods based on Bethe-Kikuchi approximations [1] and tree-reweighted upper bounds [24]. We adopt the simplest formulation of both methods in which the regions (or junction graph nodes) are defined as the edges $E$. Since loopy belief propagation failed to converge for the complete graph, we implemented the convergent double-loop algorithm of [10].

The results of the experiments are summarized in Fig. 2. The plots on the left and right show that the estimate of the log-partition function, for the most part, moves to the exact solution as the graph is partitioned into smaller and smaller pieces. Both Bethe-Kikuchi approximations and tree-reweighted upper bounds provide good approximations to the grid model. Indeed, the former recovers the log-partition function almost perfectly. However, these approximations break down as soon as they encounter a dense, frustrated model. This is consistent with the results observed in other experiments [9, 24]. The SMC algorithms proposed here and in [9], by contrast, produce significantly improved estimates of the mean statistics. It is surprising that we achieve similar performance with hot coupling [9], given that we do not exploit the tractability of sum-product messages in the Ising model (which would offer guaranteed improvements due to the Rao-Blackwell theorem).

## 6    Conclusions and discussion

We presented a sequential Monte Carlo algorithm in which each artificial distribution is the solution to a conditionally-specified mean field optimization problem. We believe that the extra expense of nonlinear optimization at each step may be warranted in the long run as our method holds promise in solving more difficult inference problems, problems where Monte Carlo and variational methods alone perform poorly. We hypothesize that our approach is superior methods that "prune" constraints on factors, but further exploration in other problems is needed to verify this theory.

**Beyond mean field.** As noted in [22], naive mean field implies complete factorizability, which is not necessary under the Ising model. A number of refinements are possible. However, this is not a research direction we will pursue. Bethe-Kikuchi approximations based on junction graphs have many merits, but they cannot be considered candidates for our framework because they produce

estimates of local mean statistics without defining a joint distribution. Tree-reweighted upper bounds are appealing because they tend to be underconfident, but again we have the same difficulty.

**Extending to other members of the exponential family.** In general, the joint is not available in analytic form given expressions for the conditionals, but there are still some encouraging signs. For one, we can use Brook's lemma [3, Sec. 2] to derive an expression for the importance weights that does not involve the joint. Furthermore, conditions for guaranteeing the validity of conditional densities have been extensively studied in multivariate [2] and spatial statistics [3].

## Acknowledgments

We are indebted to Arnaud Doucet and Firas Hamze for invaluable discussions, to Martin Wainwright for providing his code, and to the Natural Sciences and Engineering Research Council of Canada for their support.

## References

[1] S. M. Aji and R. J. McEliece. The Generalized distributive law and free energy minimization. In *Proceedings of the 39th Allerton Conference*, pages 672–681, 2001.

[2] B. Arnold, E. Castillo, and J.-M. Sarabia. *Conditional Specification of Statistical Models*. Springer, 1999.

[3] J. Besag. Spatial interaction and the statistical analysis of lattice systems. *J. Roy. Statist. Soc., Ser. B*, 36:192–236, 1974.

[4] J. Besag. Comment to "Conditionally specified distributions". *Statist. Sci.*, 16:265–267, 2001.

[5] W. Buntine and A. Jakulin. Applying discrete PCA in data analysis. In *Uncertainty in Artificial Intelligence*, volume 20, pages 59–66, 2004.

[6] N. de Freitas, P. Højen-Sørensen, M. I. Jordan, and S. Russell. Variational MCMC. In *Uncertainty in Artificial Intelligence*, volume 17, pages 120–127, 2001.

[7] P. del Moral, A. Doucet, and A. Jasra. Sequential Monte Carlo samplers. *J. Roy. Statist. Soc., Ser. B*, 68:411–436, 2006.

[8] Z. Ghahramani and M. J. Beal. Variational inference for Bayesian mixtures of factor analysers. In *Advances in Neural Information Processing Systems*, volume 12, pages 449–455, 1999.

[9] F. Hamze and N. de Freitas. Hot Coupling: a particle approach to inference and normalization on pairwise undirected graphs. *Advances in Neural Information Processing Systems*, 18:491–498, 2005.

[10] T. Heskes, K. Albers, and B. Kappen. Approximate inference and constrained optimization. In *Uncertainty in Artificial Intelligence*, volume 19, pages 313–320, 2003.

[11] C. Jarzynski. Nonequilibrium equality for free energy differences. *Phys. Rev. Lett.*, 78:2690–2693, 1997.

[12] M. Jerrum and A. Sinclair. The Markov chain Monte Carlo method: an approach to approximate counting and integration. In *Approximation Algorithms for NP-hard Problems*, pages 482–520. PWS Pubs., 1996.

[13] G. Kitagawa. Monte Carlo filter and smoother for non-Gaussian nonlinear state space models. *J. Comput. Graph. Statist.*, 5:1–25, 1996.

[14] P. Muyan and N. de Freitas. A blessing of dimensionality: measure concentration and probabilistic inference. In *Proceedings of the 19th Workshop on Artificial Intelligence and Statistics*, 2003.

[15] R. M. Neal. Annealed importance sampling. *Statist. and Comput.*, 11:125–139, 2001.

[16] M. Newman and G. Barkema. *Monte Carlo Methods in Statistical Physics*. Oxford Univ. Press, 1999.

[17] M. Opper and D. Saad, editors. *Advanced Mean Field Methods, Theory and Practice*. MIT Press, 2001.

[18] C. P. Robert and G. Casella. *Monte Carlo Statistical Methods*. Springer, 2nd edition, 2004.

[19] M. N. Rosenbluth and A. W. Rosenbluth. Monte Carlo calculation of the average extension of molecular chains. *J. Chem. Phys.*, 23:356–359, 1955.

[20] J. S. Sadowsky and J. A. Bucklew. On large deviations theory and asymptotically efficient Monte Carlo estimation. *IEEE Trans. Inform. Theory*, 36:579–588, 1990.

[21] L. K. Saul, T. Jaakola, and M. I. Jordan. Mean field theory for sigmoid belief networks. *J. Artificial Intelligence Res.*, 4:61–76, 1996.

[22] L. K. Saul and M. I. Jordan. Exploiting tractable structures in intractable networks. In *Advances in Neural Information Processing Systems*, volume 8, pages 486–492, 1995.

[23] E. B. Sudderth, A. T. Ihler, W. T. Freeman, and A. S. Willsky. Nonparametric belief propagation. In *Computer Vision and Pattern Recognition,*, volume I, pages 605–612, 2003.

[24] M. J. Wainwright, T. S. Jaakkola, and A. S. Willsky. A new class of upper bounds on the log partition function. *IEEE Trans. Inform. Theory*, 51:2313–2335, 2005.

[25] M. J. Wainwright and M. I. Jordan. Graphical models, exponential families, and variational inference. Technical report, EECS Dept., University of California, Berkeley, 2003.

[26] W. Wiegerinck. Variational approximations between mean field theory and the junction tree algorithm. In *Uncertainty in Artificial Intelligence*, volume 16, pages 626–633, 2000.
